# A Simple and Fast Neural Network Approach to Stereovision

**Rolf D. Henkel**
Institute of Theoretical Physics
University of Bremen
P.O. Box 330 440, D-28334 Bremen
http://axon.physik.uni-bremen.de/~rdh

## Abstract

A neural network approach to stereovision is presented based on aliasing effects of simple disparity estimators and a fast coherence-detection scheme. Within a single network structure, a dense disparity map with an associated validation map and, additionally, the fused cyclopean view of the scene are available. The network operations are based on simple, biological plausible circuitry; the algorithm is fully parallel and non-iterative.

## 1 Introduction

Humans experience the three-dimensional world not as it is seen by either their left or right eye, but from a position of a virtual cyclopean eye, located in the middle between the two real eye positions. The different perspectives between the left and right eyes cause slight relative displacements of objects in the two retinal images (disparities), which make a simple superposition of both images without diplopia impossible. Proper fusion of the retinal images into the cyclopean view requires the registration of both images to a common coordinate system, which in turn requires calculation of disparities for all image areas which are to be fused.

### 1.1 The Problems with Classical Approaches

The estimation of disparities turns out to be a difficult task, since various random and systematic image variations complicate this task. Several different techniques have been proposed over time, which can be loosely grouped into feature-, area-

and phase-based approaches. All these algorithms have a number of computational problems directly linked to the very assumptions inherent in these approaches.

In feature-based stereo, intensity data is first converted to a set of features assumed to be a more stable image property than the raw image intensities. Matching primitives used include zerocrossings, edges and corner points (Frisby, 1991), or higher order primitives like topological fingerprints (see for example: Fleck, 1991). Generally, the set of feature-classes is *discrete*, causing the two primary problems of feature-based stereo algorithms: the famous "false-matches"-problem and the problem of missing disparity estimates.

False matches are caused by the fact that a single feature in the left image can potentially be matched with every feature of the same class in the right image. This problem is basic to all feature-based stereo algorithms and can only be solved by the introduction of additional constraints to the solution. In conjunction with the extracted features these constraints define a complicated error measure which can be minimized by cooperative processes (Marr, 1979) or by direct (Ohta, 1985) or stochastic search techniques (Yuille, 1991). While cooperative processes and stochastic search techniques can be realized easily on a neural basis, it is not immediately clear how to implement the more complicated algorithmic structures of direct search techniques neuronally. Cooperative processes and stochastic search techniques turn out to be slow, needing many iterations to converge to a local minimum of the error measure.

The requirement of features to be a stable image property causes the second problem of feature-based stereo: stable features can only be detected in a fraction of the whole image area, leading to missing disparity estimates for most of the image area. For those image parts, disparity estimates can only be guessed.

Dense disparity maps can be obtained with area-based approaches, where a suitable chosen correlation measure is maximized between small image patches of the left and right view. However, a neuronally plausible implementation of this seems to be not readily available. Furthermore, the maximization turns out to be a computationally expensive process, since extensive search is required in configuration space.

Hierarchical processing schemes can be utilized for speed-up, by using information obtained at coarse spatial scales to restrict searching at finer scales. But, for general image data, it is not guaranteed that the disparity information obtained at some coarse scale is valid. The disparity data might be wrong, might have a different value than at finer scales, or might not be present at all. Furthermore, by processing data from coarse to fine spatial scales, hierarchical processing schemes are intrinsically sequential. This creates additional algorithmic overhead which is again difficult to realize with neuronal structures.

The same comments apply to phase-based approaches, where a locally extracted Fourier-phase value is used for matching. Phase values are only defined modulo $2\pi$, and this wrap-around makes the use of hierarchical processing essential for these types of algorithms. Moreover, since data is analyzed in different spatial frequency channels, it is nearly certain that some phase values will be undefined at intermediate scales, due to missing signal energy in this frequency band (Fleet, 1993). Thus, in addition to hierarchical processing, some kind of exception handling is needed with these approaches.

## 2    Stereovision by Coherence Detection

In summary, classical approaches to stereovision seem to have difficulties with the
fast calculation of dense disparity-maps, at least with plausible neural circuitry.
In the following, a neural network implementation will be described which solves
this task by using simple disparity estimators based on motion-energy mechanisms
(Adelson, 1985; Qian, 1997), closely resembling responses of complex cells in visual
cortex (DeAngelis, 1991). Disparity units of these type belong to a class of disparity
estimators which can be derived from optical flow methods (Barron, 1994). Clearly,
disparity calculations and optical flow estimation share many similarities. The two
stereo views of a (static) scene can be considered as two time-slices cut out of
the space-time intensity pattern which would be recorded by an imaginary camera
moving from the position of the left to the position of the right eye. However,
compared to optical flow, disparity estimation is complicated by the fact that only
two discrete "time"-samples are available, namely the images of the left and right
view positions.

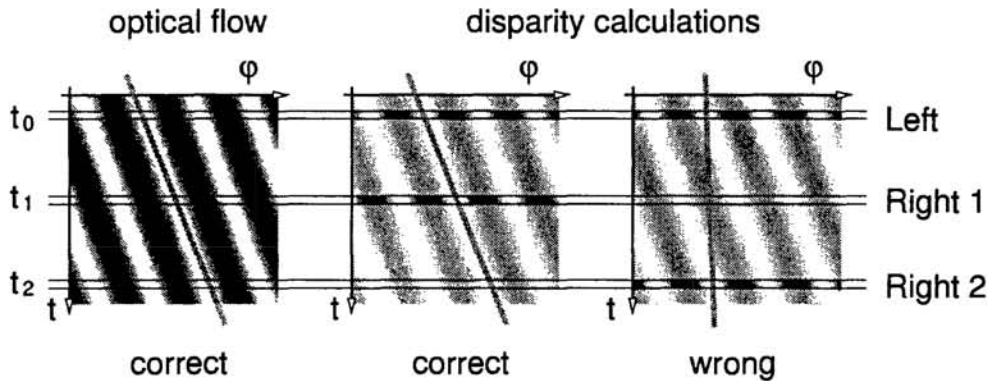

Figure 1: The velocity of an image patch manifests itself as principal texture direc-
tion in the space-time flow field traced out by the intensity pattern in time (left).
Sampling such flow patterns at discrete times can create aliasing-effects which lead
to wrong estimates. If one is using optical flow estimation techniques for disparity
calculations, this problem is always present.

For an explanation consider Fig. 1. A surface patch shifting over time traces out
a certain flow pattern. The principal texture direction of this flow indicates the
relative velocity of the image patch (Fig. 1, left). Sampling the flow pattern only
at discrete time points, the shift between two "time-samples" can be estimated
without ambiguity provided the shift is not too large (Fig. 1, middle). However, if a
certain limit is exceeded, it becomes impossible to estimate the shift correctly, given
the data (Fig. 1, right). This is a simple aliasing-effect in the "time"-direction; an
everyday example can be seen as motion reversal in movies.

In the case of stereovision, aliasing-effects of this type are always present, and they
limit the range of disparities a simple disparity unit can estimate. Sampling theory
gives a relation between the maximal spatial wavevector $k_{\max}^{\varphi}$ (or, equivalently, the
minimum spatial wavelength $\lambda_{\min}^{\varphi}$) present in the data and the largest disparity
which can be estimated reliably (Henkel, 1997):

$$|d| < \frac{\pi}{k_{\max}^{\varphi}} = \frac{1}{2}\lambda_{\min}^{\varphi} . \qquad (1)$$

A well-known example of the size-disparity scaling expressed in equation (1) is found in the context of the spatial frequency channels assumed to exist in the visual cortex. Cortical cells respond to spatial wavelengths down to about half their peak wavelength $\lambda_{opt}$; therefore, they can estimate reliable only disparities less than $1/4\,\lambda_{opt}$. This is known as Marr's quarter-cycle limit (Blake, 1991).

Equation (1) immediately suggests a way to extend the limited working range of disparity estimators: a spatial smoothing of the image data before or during disparity calculation reduces $k^{\varphi}_{max}$, and in turn increases the disparity range. However, spatial smoothing reduces also the spatial resolution of the resulting disparity map. Another way of modifying the usable range of disparity estimators is the application of a fixed preshift to the input data before disparity calculation. This would require prior knowledge of the correct preshift to be applied, which is a nontrivial problem. One could resort to hierarchical coarse-to-fine schemes, but the difficulties with hierarchical schemes have already been elaborated.

The aliasing effects discussed are a general feature of sampling visual space with only two eyes; instead of counteracting, one can exploit them in a simple coherence-detection scheme, where the multi-unit activity in stacks of disparity detectors tuned to a common view direction is analyzed.

Assuming that all disparity units $i$ in a stack have random preshifts or presmoothing applied to their input data, these units will have different, but slightly overlapping working ranges $D_i = [d_i^{min}, d_i^{max}]$ for valid disparity estimates. An object with true disparity $d$, seen in the common view direction of such a stack, will therefore split the stack into two disjunct classes: the class $C$ of estimators with $d \in D_i$ for all $i \in C$, and the rest of the stack, $\overline{C}$, with $d \notin D_i$. All disparity estimators $\in C$ will code more or less the true disparity $d_i \approx d$, but the estimates of units belonging to $\overline{C}$ will be subject to the random aliasing effects discussed, depending in a complicated way on image content and disparity range $D_i$ of the unit.

We will thus have $d_i \approx d \approx d_j$ whenever units $i$ and $j$ belong to $C$, and random relationships otherwise. A simple coherence detection within each stack, i.e. searching for all units with $d_i \approx d_j$ and extracting the largest cluster found, will be sufficient to single out $C$. The true disparity $d$ in the view direction of the stack can be simply estimated as an average over all coherently coding units:

$$d \approx \langle d_i \rangle_{i \in C}\ .$$

## 3   Neural Network Implementation

Repeating this coherence detection scheme in every view direction results in a fully parallel network structure for disparity calculation. Neighboring disparity stacks responding to different view directions estimate disparity values independently from each other, and within each stack, disparity units operate independently from each other. Since coherence detection is an opportunistic scheme, extensions of the basic algorithm to multiple spatial scales and combinations of different types of disparity estimators are trivial. Additional units are simply included in the appropriate coherence stacks. The coherence scheme will combine only the information from the coherently coding units and ignore the rest of the data. For this reason, the scheme also turns out to be extremely robust against single-unit failures.

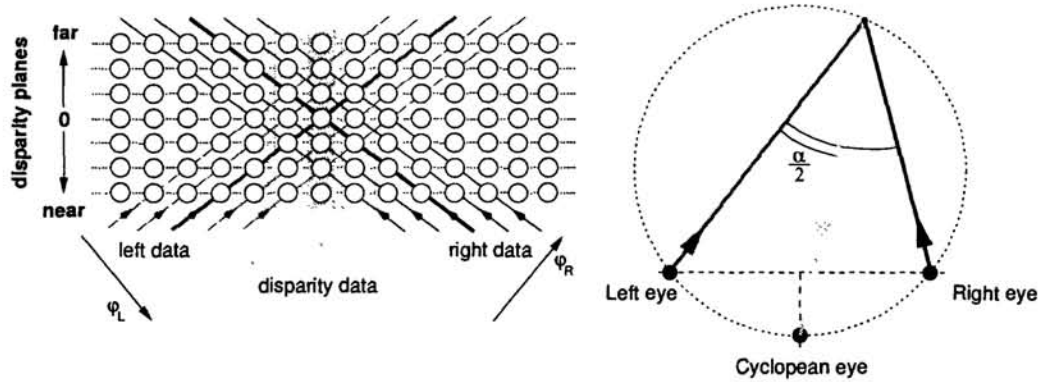

Figure 2: The network structure for a single horizontal scan-line (left). The view directions of the disparity stacks split the angle between the left and right lines of sight in the network and 3D-space in half, therefore analyzing space along the cyclopean view directions (right).

In the current implementation (Fig. 2), disparity units at a single spatial scale are arranged into horizontal disparity layers. Left and right image data is fed into this network along diagonally running data lines. This causes every disparity layer to receive the stereo data with a certain fixed preshift applied, leading to the required, slightly different working-ranges of neighboring layers. Disparity units stacked vertically above each other are collected into a single disparity stack which is then analyzed for coherent activity.

## 4   Results

The new stereo network performs comparable on several standard test image sets (Fig. 3). The calculated disparity maps are similar to maps obtained by classical area-based approaches, but they display subpixel-precision. Since no smoothing or regularization is performed by the coherence-based stereo algorithm, sharp disparity edges can be observed at object borders.

Within the network, a simple validation map is available locally. A measure of local

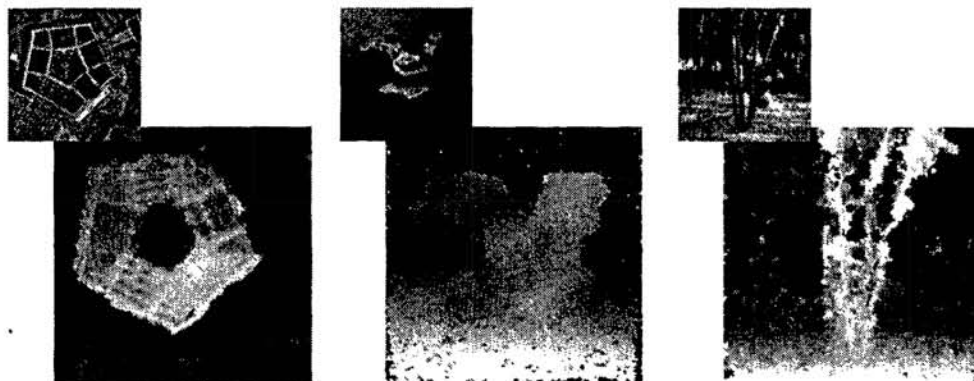

Figure 3: Disparity maps for some standard test images (small insets), calculated by the coherence-based stereo algorithm.

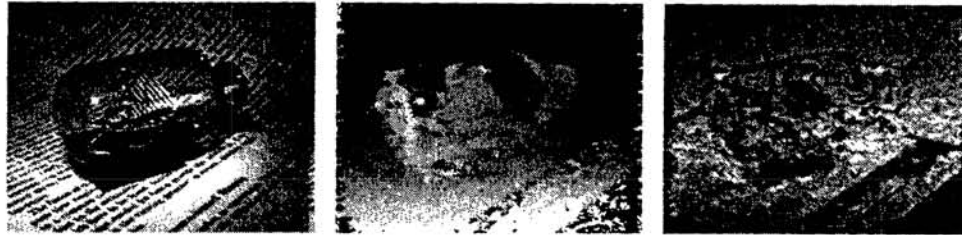

Figure 4: The performance of coherence-based stereo on a difficult scene with specular highlights, transparency and repetitive structures (left). The disparity map (middle) is dense and correct, except for a few structure-less image regions. These regions, as well as most object borders, are indicated in the validation map (right) with a low [dark] validation count.

coherence can be obtained by calculating the relative number of coherently acting disparity units in each stack, i.e. by calculating the ratio $N(\mathcal{C})/\mathcal{N}(\mathcal{C}\cup\overline{\mathcal{C}})$, where $N(\mathcal{C})$ is the number of units in class $\mathcal{C}$. In most cases, this validation map clearly marks image areas where the disparity calculations failed (for various reasons, notably at occlusions caused by object borders, or in large structure-less image regions, where no reliable matching can be obtained — compare Fig 4).

Close inspection of disparity and validation maps reveals that these image maps are not aligned with the left or the right view of the scene. Instead, both maps are registered with the cyclopean view. This is caused by the structural arrangement of data lines and disparity stacks in the network. Reprojecting data lines and stacks back into 3D-space shows that the stacks analyze three-dimensional space along lines splitting the angle between the left and right view directions in half. This is the cyclopean view direction as defined by (Hering, 1879).

It is easy to obtain the cyclopean view of the scene itself. With $I_i^L$ and $I_i^R$ denoting the left and right input data at the position of disparity-unit $i$, a summation over all coherently coding disparity units in a stack, i.e.,

$$I^{\mathcal{C}} = \left\langle I_i^L + I_i^R \right\rangle_{i\in\mathcal{C}} ,$$

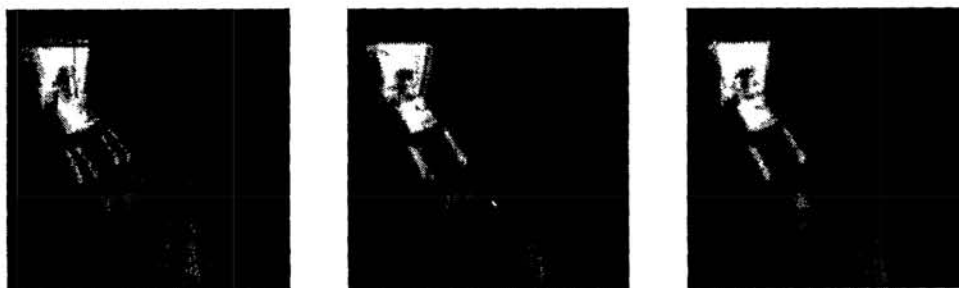

Figure 5: A simple superposition of the left and right stereo images results in diplopia (left). By using a vergence system, the two stereo images can be aligned better (middle), but diplopia is still prominent in most areas of the visual field. The fused cyclopean view of the scene (left) was calculated by the coherence-based stereo network.

gives the image intensity $I^C$ in the cyclopean view-direction of this stack. Collecting $I^C$ from all disparity stacks gives the complete cyclopean view as the third co-registered map of the network (Fig 5).

## Acknowledgements

Thanks to Helmut Schwegler and Robert P. O'Shea for interesting discussions. Image data courtesy of G. Medoni, UCS Institute for Robotics & Intelligent Systems, B. Bolles, AIC, SRI International, and G. Sommer, Kiel Cognitive Systems Group, Christian-Albrechts-Universität Kiel. An internet-based implementation of the algorithm presented in this paper is available at http://axon.physik.uni-bremen.de/~rdh/online_calc/stereo/.

## References

Adelson, E.H. & Bergen, J.R. (1985): Spatiotemporal Energy Models for the Perception of Motion. *J. Opt. Soc. Am.* **A2**: 284–299.

Barron, J.L., Fleet, D.J. & Beauchemin, S.S. (1994): Performance of Optical Flow Techniques. *Int. J. Comp. Vis.* **12**: 43–77.

Blake, R. & Wilson, H.R. (1991): Neural Models of Stereoscopic Vision. *TINS* **14**: 445–452.

DeAngelis, G.C., Ohzawa, I. & Freeman, R.D. (1991): Depth is Encoded in the Visual Cortex by a Specialized Field Structure. *Nature* **11**: 156–159.

Fleck, M.M. (1991): A Topological Stereo Matcher. *Int. J. of Comp. Vis.* **6**: 197–226.

Fleet, D.J. & Jepson, A.D. (1993): Stability of Phase Information. *IEEE PAMI* **2**: 333-340.

Frisby, J.P. & and S. B. Pollard, S.B. (1991): Computational Issues in Solving the Stereo Correspondence Problem. eds. M.S. Landy and J. A. Movshon, *Computational Models of Visual Processing*, pp. 331, MIT Press, Cambridge 1991.

Henkel, R.D. (1997): Fast Stereovision by Coherence Detection, in *Proc. of CAIP'97, Kiel*, LCNS 1296, eds. G. Sommer, K. Daniilidis and J. Pauli, pp. 297, LCNS 1296, Springer, Heidelberg 1997.

E. Hering (1879): Der Raumsinn und die Bewegung des Auges, in Handbuch der Psychologie, ed. L. Hermann, Band 3, Teil 1, Vogel, Leipzig 1879.

Marr, D. & Poggio, T. (1979): A Computational Theory of Human Stereo Vision. *Proc. R. Soc. Lond.* **B 204**: 301–328.

Ohta, Y, & Kanade, T. (1985): Stereo by Intra- and Inter-scanline Search using dynamic programming. *IEEE PAMI* **7**: 139–154.

Qian, N. & Zhu, Y. (1997): Physiological Computation of Binocular Disparity, to appear in *Vision Research*.

Yuille, A.L., Geiger, D. & Bülthoff, H.H. (1991): Stereo Integration, Mean Field Theory and Psychophysics. *Network* **2**: 423–442.